# Training Conditional Random Fields for Maximum Labelwise Accuracy

**Samuel S. Gross**
Computer Science Department
Stanford University
Stanford, CA, USA
ssgross@cs.stanford.edu

**Olga Russakovsky**
Computer Science Department
Stanford University
Stanford, CA, USA
olga@cs.stanford.edu

**Chuong B. Do**
Computer Science Department
Stanford University
Stanford, CA, USA
chuongdo@cs.stanford.edu

**Serafim Batzoglou**
Computer Science Department
Stanford University
Stanford, CA, USA
serafim@cs.stanford.edu

## Abstract

We consider the problem of training a conditional random field (CRF) to maximize per-label predictive accuracy on a training set, an approach motivated by the principle of empirical risk minimization. We give a gradient-based procedure for minimizing an arbitrarily accurate approximation of the empirical risk under a Hamming loss function. In experiments with both simulated and real data, our optimization procedure gives significantly better testing performance than several current approaches for CRF training, especially in situations of high label noise.

## 1 Introduction

Sequence labeling, the task of assigning labels $\mathbf{y} = y_1, ..., y_L$ to an input sequence $\mathbf{x} = x_1, ..., x_L$, is a machine learning problem of great theoretical and practical interest that arises in diverse fields such as computational biology, computer vision, and natural language processing. Conditional random fields (CRFs) are a class of discriminative probabilistic models designed specifically for sequence labeling tasks [1]. CRFs define the conditional distribution $P_{\mathbf{w}}(\mathbf{y} \mid \mathbf{x})$ as a function of features relating labels to the input sequence.

Ideally, training a CRF involves finding a parameter set $\mathbf{w}$ that gives high accuracy when labeling new sequences. In some cases, however, simply finding parameters that give the best possible accuracy on training data (known as *empirical risk minimization* [2]) can be difficult. In particular, if we wish to minimize Hamming loss, which measures the number of incorrect labels, gradient-based optimization methods cannot be applied directly.[1] Consequently, surrogate optimization problems, such as maximum likelihood or maximum margin training, are solved instead.

In this paper, we describe a training procedure that addresses the problem of minimizing empirical per-label risk for CRFs. Specifically, our technique attempts to minimize a smoothed approximation of the Hamming loss incurred by the maximum expected accuracy decoding (i.e., posterior decoding) algorithm on the training set. The degree of approximation is controlled by a parameterized function $Q(\cdot)$ which trades off between the accuracy of the approximation and the smoothness of the objective. In the limit as $Q(\cdot)$ approaches the step function, the optimization objective converges to the empirical risk minimization criterion for Hamming loss.

## 2 Preliminaries

### 2.1 Definitions

Let $\mathcal{X}^L$ denote an input space of all possible input sequences, and let $\mathcal{Y}^L$ denote an output space of all possible output labels. Furthermore, for a pair of consecutive labels $y_{j-1}$ and $y_j$, an input sequence $\mathbf{x}$, and a label position $j$, let $\mathbf{f}(y_{j-1}, y_j, \mathbf{x}, j) \in \mathbb{R}^n$ be a vector-valued function; we call $\mathbf{f}$ the *feature mapping* of the CRF.

A conditional random field (CRF) defines the conditional probability of a labeling (or parse) $\mathbf{y}$ given an input sequence $\mathbf{x}$ as

$$P_{\mathbf{w}}(\mathbf{y} \mid \mathbf{x}) = \frac{\exp\left(\sum_{j=1}^{L} \mathbf{w}^T \mathbf{f}(y_{j-1}, y_j, \mathbf{x}, j)\right)}{\sum_{\mathbf{y}' \in \mathcal{Y}^L} \exp\left(\sum_{j=1}^{L} \mathbf{w}^T \mathbf{f}(y'_{j-1}, y'_j, \mathbf{x}, j)\right)} = \frac{\exp\left(\mathbf{w}^T \mathbf{F}_{1,L}(\mathbf{x}, \mathbf{y})\right)}{\mathcal{Z}(\mathbf{x})}, \qquad (1)$$

where we define the *summed feature mapping*, $\mathbf{F}_{a,b}(\mathbf{x}, \mathbf{y}) = \sum_{j=a}^{b} \mathbf{f}(y_{j-1}, y_j, \mathbf{x}, j)$, and where the *partition function* $\mathcal{Z}(\mathbf{x}) = \sum_{\mathbf{y}'} \exp\left(\mathbf{w}^T \mathbf{F}_{1,L}(\mathbf{x}, \mathbf{y}')\right)$ ensures that the distribution is normalized for any set of model parameters $\mathbf{w}$.[2]

### 2.2 Maximum *a posteriori* vs. maximum expected accuracy parsing

Given a CRF with parameters $\mathbf{w}$, the sequence labeling task is to determine values for the labels $\mathbf{y}$ of a new input sequence $\mathbf{x}$. One approach is to choose the most likely, or maximum *a posteriori*, labeling, $\arg\max_{\mathbf{y}} P_{\mathbf{w}}(\mathbf{y} \mid \mathbf{x})$. This can be computed efficiently using the Viterbi algorithm.

An alternative approach, which seeks to maximize the per-label accuracy of the prediction rather than the joint probability of the entire parse, chooses the most likely (i.e., highest posterior probability) value for each label separately. Note that

$$\arg\max_{\mathbf{y}} \sum_{j=1}^{L} P_{\mathbf{w}}(y_j \mid \mathbf{x}) = \arg\max_{\mathbf{y}} \mathbb{E}_{\mathbf{y}'}\left[\sum_{j=1}^{L} 1\{y'_j = y_j\}\right] \qquad (2)$$

where $1\{condition\}$ denotes the usual indicator function whose value is 1 when $condition$ is true and 0 otherwise, and where the expectation is taken with respect to the conditional distribution $P_{\mathbf{w}}(\mathbf{y}' \mid \mathbf{x})$. From this, we see that maximum expected accuracy parsing chooses the parse with the maximum expected number of correct labels.

In practice, maximum expected accuracy parsing often yields more accurate results than Viterbi parsing (on a per-label basis) [3, 4, 5]. Here, we restrict our focus to maximum expected accuracy parsing procedures and seek training criteria which optimize the performance of a CRF-based maximum expected accuracy parser.

## 3 Training conditional random fields

Usually, CRFs are trained in the batch setting, where a complete set $\mathcal{D} = \{(\mathbf{x}^{(t)}, \mathbf{y}^{(t)})\}_{t=1}^{m}$ of training examples is available up front. In this case, training amounts to numerical optimization of a fixed objective function $\mathcal{R}(\mathbf{w} : \mathcal{D})$. A good objective function is one whose optimal value leads to parameters that perform well, in an application-dependent sense, on previously unseen testing examples. While this can be difficult to achieve without knowing the contents of the testing set, one can, under certain conditions, guarantee that the accuracy of a learned CRF on an unseen testing set is probably not much worse than its accuracy on the training set.

In particular, when assuming independently and identically distributed (i.i.d.) training and testing examples, there exists a probabilistic bound on the difference between empirical risk and generalization error [2]. As long as enough training data are available (relative to model complexity), strong training set performance will imply, with high probability, similarly strong testing set performance. Unfortunately, minimizing empirical risk for a CRF is a very difficult task. Loss functions based on usual notions of per-label accuracy (such as Hamming loss) are typically not only nonconvex but also not amenable to optimization by methods that make use of gradient information.

In this section, we briefly describe three previous approaches for CRF training which optimize surrogate loss functions in lieu of the empirical risk. Then, we consider a new method for gradient-based CRF training oriented more directly toward optimizing predictive performance on the training set. Our method minimizes an arbitrarily accurate approximation of empirical risk, where the loss function is defined as the number of labels predicted incorrectly by maximum expected accuracy parsing.

## 3.1 Previous objective functions

### 3.1.1 Conditional log-likelihood

Conditional log-likelihood is the most commonly used objective function for training conditional random fields. In this criterion, the loss suffered for a training example $(\mathbf{x}^{(t)}, \mathbf{y}^{(t)})$ is the negative log probability of the true parse according to the model, plus a regularization term:

$$\mathcal{R}_{\text{CLL}}(\mathbf{w} : \mathcal{D}) = C||\mathbf{w}||^2 - \sum_{t=1}^{m} \log P_{\mathbf{w}}(\mathbf{y}^{(t)} \mid \mathbf{x}^{(t)}) \tag{3}$$

The convexity and differentiability of conditional log-likelihood ensure that gradient-based optimization procedures (e.g., conjugate gradient or L-BFGS [6]) will not converge to suboptimal local minima of the objective function.

However, there is no guarantee that the parameters obtained by conditional log-likelihood training will lead to the best per-label predictive accuracy, even on the training set. For one, maximum likelihood training explicitly considers only the probability of exact training parses. Other parses, even highly accurate ones, are ignored except insofar as they share common features with the exact parse. In addition, the log-likelihood of a parse is largely determined by the sections which are most difficult to correctly label. This can be a weakness in problems with significant label noise (i.e., incorrectly labeled training examples).

### 3.1.2 Pointwise conditional log likelihood

Kakade et al. investigated an alternative nonconvex training objective for CRFs [7, 8] which considers separately the posterior label probabilities at each position of each training sequence. In this approach, one maximizes not the probability of an entire parse, but instead the product of the posterior probabilities (or equivalently, sum of log posteriors) for each predicted label:

$$\mathcal{R}_{\text{pointwise}}(\mathbf{w} : \mathcal{D}) = C||\mathbf{w}||^2 - \sum_{t=1}^{m} \sum_{j=1}^{L} \log P_{\mathbf{w}}(y_j^{(t)} \mid \mathbf{x}^{(t)}) \tag{4}$$

By using pointwise posterior probabilities, this objective function takes into account suboptimal parses and focuses on finding a model whose posteriors match well with the training labels, even though the model may not provide a good fit for the training data as a whole.

Nevertheless, pointwise logloss is fundamentally quite different from Hamming loss. A training procedure based on pointwise log likelihood, for example, would prefer to reduce the posterior probability for a correct label from 0.6 to 0.4 in return for improving the posterior probability for a hopelessly incorrect label from 0.0001 to 0.01. Thus, the objective retains the difficulties of the regular conditional log likelihood when dealing with difficult-to-classify outlier labels.

### 3.1.3 Maximum margin training

The notion of Hamming distance is incorporated directly in the maximum margin training procedures of Taskar et al. [9]:

$$\mathcal{R}_{\text{max margin}}(\mathbf{w} : \mathcal{D}) = C||\mathbf{w}||^2 + \sum_{t=1}^{m} \max \left( 0, \max_{\mathbf{y} \in \mathcal{Y}^L} \left( \Delta(\mathbf{y}, \mathbf{y}^{(t)}) - \mathbf{w}^T \delta \mathbf{F}_{1,L}(\mathbf{x}^{(t)}, \mathbf{y}) \right) \right), \tag{5}$$

and Tsochantaridis et al. [10].

$$\mathcal{R}_{\text{max margin}}(\mathbf{w} : \mathcal{D}) = C||\mathbf{w}||^2 + \sum_{t=1}^{m} \max \left( 0, \max_{\mathbf{y} \in \mathcal{Y}^L} \Delta(\mathbf{y}, \mathbf{y}^{(t)}) \left( 1 - \mathbf{w}^T \delta \mathbf{F}_{1,L}(\mathbf{x}^{(t)}, \mathbf{y}) \right) \right). \tag{6}$$

Here, $\Delta(\mathbf{y}, \mathbf{y}^{(t)})$ denotes the Hamming distance between $\mathbf{y}$ and $\mathbf{y}^{(\mathbf{t})}$, and $\delta \mathbf{F}_{1,L}(\mathbf{x}^{(t)}, \mathbf{y}) = \mathbf{F}_{1,L}(\mathbf{x}^{(t)}, \mathbf{y}^{(t)}) - \mathbf{F}_{1,L}(\mathbf{x}^{(t)}, \mathbf{y})$. In the former formulation, loss is incurred when the Hamming distance between the correct parse $\mathbf{y}^{(t)}$ and a candidate parse $\mathbf{y}$ exceeds the obtained classification

margin between $\mathbf{y}^{(t)}$ and $\mathbf{y}$. In the latter formulation, the amount of loss for a margin violation scales linearly with the Hamming distance betweeen $\mathbf{y}^{(t)}$ and $\mathbf{y}$.

Both cases lead to convex optimization problems in which the loss incurred for a particular training example is an upper bound on the Hamming loss between the correct parse and its highest scoring alternative. In practice, however, this upper bound can be quite loose; thus, parameters obtained via a maximum margin framework may be poor minimizers of empirical risk.

### 3.2 Training for maximum labelwise accuracy

In each of the likelihood-based or margin-based objective functions introduced in the previous subsections, difficulties arose due to the mismatch between the chosen objective function and our notion of empirical risk as defined by Hamming loss. In this section, we demonstrate how to construct a smooth objective function for maximum expected accuracy parsing which more closely approximates our desired notion of empirical risk.

#### 3.2.1 The labelwise accuracy objective function

Consider the following objective function,

$$\mathcal{R}(\mathbf{w} : \mathcal{D}) = \sum_{t=1}^{m} \sum_{j=1}^{L} 1 \left\{ y_j^{(t)} = \arg\max_{y_j} P_{\mathbf{w}}(y_j \mid \mathbf{x}^{(t)}) \right\}. \tag{7}$$

Maximizing this objective is equivalent to minimizing empirical risk under the Hamming loss (i.e., the number of mispredicted labels). To obtain a smooth approximation to this objective function, we can express the condition that the algorithm predicts the correct label for $y_j^{(t)}$ in terms of the posterior probabilities of correct and incorrect labels as

$$P_{\mathbf{w}}(y_j^{(t)} \mid \mathbf{x}^{(t)}) - \max_{y_j \neq y_j^{(t)}} P_{\mathbf{w}}(y_j \mid \mathbf{x}^{(t)}) > 0. \tag{8}$$

Substituting equation (8) back into equation (7) and replacing the indicator function with a generic function $Q(\cdot)$, we obtain

$$\mathcal{R}_{\text{labelwise}}(\mathbf{w}) = \sum_{t=1}^{m} \sum_{j=1}^{L} Q \left( P_{\mathbf{w}}(y_j^{(t)} \mid \mathbf{x}^{(t)}) - \max_{y_j \neq y_j^{(t)}} P_{\mathbf{w}}(y_j \mid \mathbf{x}^{(t)}) \right). \tag{9}$$

When $Q(\cdot)$ is chosen to be the indicator function, $Q(x) = 1\{x > 0\}$, we recover the original objective. By choosing a nicely behaved form for $Q(\cdot)$, however, we obtain a new objective that is easier to optimize. Specifically, we set $Q(x)$ to be sigmoidal with parameter $\lambda$ (see Figure 2a):

$$Q(x; \lambda) = \frac{1}{1 + \exp(-\lambda x)}. \tag{10}$$

As $\lambda \to \infty$, $Q(x; \lambda) \to 1\{x > 0\}$, so $\mathcal{R}_{\text{labelwise}}(\mathbf{w} : \mathcal{D})$ approaches the objective function defined in (7). However, $\mathcal{R}_{\text{labelwise}}(\mathbf{w} : \mathcal{D})$ is smooth for any finite $\lambda > 0$.

Because of this, we are free to use gradient-based optimization to maximize our new objective function. As $\lambda$ get larger, the quality of our approximation of the ideal Hamming loss objective improves; however, the approximation itself also becomes less smooth and perhaps more difficult to optimize as a result. Thus, the value of $\lambda$ controls the trade-off between the accuracy of the approximation and the ease of optimization.[3]

#### 3.2.2 The labelwise accuracy objective gradient

We now present an algorithm for efficiently calculating the gradient of the approximate accuracy objective. For a fixed parameter set $\mathbf{w}$, let $\tilde{y}_j^{(t)}$ denote the label other than $y_j^{(t)}$ that has the maximum posterior probability at position $j$. Also, for notational convenience, let $\mathbf{y}_{1:j}$ denote the variables

$y_1, \ldots, y_j$. Differentiating equation (9), we compute $\nabla_{\mathbf{w}} \mathcal{R}_{\text{labelwise}}(\mathbf{w} : \mathcal{D})$ to be[4]

$$\sum_{t=1}^{m} \sum_{j=1}^{L} Q'\left(P_{\mathbf{w}}(y_j^{(t)} \mid \mathbf{x}^{(t)}) - P_{\mathbf{w}}(\tilde{y}_j^{(t)} \mid \mathbf{x}^{(t)})\right) \nabla_{\mathbf{w}}\left[P_{\mathbf{w}}(y_j^{(t)} \mid \mathbf{x}^{(t)}) - P_{\mathbf{w}}(\tilde{y}_j^{(t)} \mid \mathbf{x}^{(t)})\right]. \quad (11)$$

Using equation (1), the inner term, $P_{\mathbf{w}}(y_j^{(t)} \mid \mathbf{x}^{(t)}) - P_{\mathbf{w}}(\tilde{y}_j^{(t)} \mid \mathbf{x}^{(t)})$, is equal to

$$\frac{1}{\mathcal{Z}(\mathbf{x}^{(t)})} \cdot \sum_{\mathbf{y}_{1:L}} \left(1\{y_j = y_j^{(t)}\} - 1\{y_j = \tilde{y}_j^{(t)}\}\right) \cdot \exp\left(\mathbf{w}^T \mathbf{F}_{1,L}(\mathbf{x}^{(t)}, \mathbf{y})\right). \quad (12)$$

Applying the quotient rule allows us to compute the gradient of equation (12), whose complete form we omit for lack of space. Most of the terms involved in the gradient are easy to compute using the standard forward and backward matrices used for regular CRF inference, which we define here as

$$\alpha(i, j) = \sum_{\mathbf{y}_{1:j}} 1\{y_j = i\} \cdot \exp\left(\mathbf{w}^T \mathbf{F}_{1,j}(\mathbf{x}^{(t)}, \mathbf{y})\right) \quad (13)$$

$$\beta(i, j) = \sum_{\mathbf{y}_{j:L}} 1\{y_j = i\} \cdot \exp\left(\mathbf{w}^T \mathbf{F}_{j+1,L}(\mathbf{x}^{(t)}, \mathbf{y})\right). \quad (14)$$

The two difficult terms that do not follow from the forward and backward matrices have the form,

$$\sum_{k=1}^{L} \sum_{\mathbf{y}_{1:L}} Q'_k(\mathbf{w}) \cdot 1\{y_k = y_k^\star\} \cdot \mathbf{F}_{1,L}(\mathbf{x}^{(t)}, \mathbf{y}) \cdot \exp\left(\mathbf{w}^T \mathbf{F}_{1,L}(\mathbf{x}^{(t)}, \mathbf{y})\right), \quad (15)$$

where $Q'_j(\mathbf{w}) = Q'\left(P_{\mathbf{w}}(y_j^{(t)} \mid \mathbf{x}^{(t)}) - P_{\mathbf{w}}(\tilde{y}_j^{(t)} \mid \mathbf{x}^{(t)})\right)$ and $\mathbf{y}^\star$ is either $\mathbf{y}^{(t)}$ or $\tilde{\mathbf{y}}^{(t)}$. To efficiently compute terms of this type, we define

$$\alpha^\star(i, j) = \sum_{k=1}^{j} \sum_{\mathbf{y}_{1:j}} 1\{y_k = y_k^\star \wedge y_j = i\} \cdot Q'_k(\mathbf{w}) \cdot \exp\left(\mathbf{w}^T \mathbf{F}_{1,j}(\mathbf{x}^{(t)}, \mathbf{y})\right) \quad (16)$$

$$\beta^\star(i, j) = \sum_{k=j+1}^{L} \sum_{\mathbf{y}_{j:L}} 1\{y_k = y_k^\star \wedge y_j = i\} \cdot Q'_k(\mathbf{w}) \cdot \exp\left(\mathbf{w}^T \mathbf{F}_{j+1,L}(\mathbf{x}^{(t)}, \mathbf{y})\right). \quad (17)$$

Like the forward and backward matrices, $\alpha^\star(i, j)$ and $\beta^\star(i, j)$ may be calculated via dynamic programming. In particular, we have the base cases $\alpha^\star(i, 1) = 1\{i = y_1^\star\} \cdot \alpha(i, 1) \cdot Q'_1(\mathbf{w})$ and $\beta^\star(i, L) = 0$. The remaining entries are given by the following recurrences:

$$\alpha^\star(i, j) = \sum_{i'} \left(\alpha^\star(i', j-1) + 1\{i = y_j^\star\} \cdot \alpha(i', j-1) \cdot Q'_j(\mathbf{w})\right) \cdot e^{\mathbf{w}^T \mathbf{f}(i', i, \mathbf{x}^{(t)}, j)} \quad (18)$$

$$\beta^\star(i, j) = \sum_{i'} \left(\beta^\star(i', j+1) + 1\{i' = y_{j+1}^\star\} \cdot \beta(i', j+1) \cdot Q'_{j+1}(\mathbf{w})\right) \cdot e^{\mathbf{w}^T \mathbf{f}(i, i', \mathbf{x}^{(t)}, j+1)}. \quad (19)$$

It follows that equation (15) is equal to

$$\sum_{j=1}^{L} \sum_{i'} \sum_{i} \mathbf{f}(i', i, \mathbf{x}^{(t)}, j) \cdot \exp\left(\mathbf{w}^T \mathbf{f}(i', i, \mathbf{x}^{(t)}, j)\right) \cdot (A + B), \quad (20)$$

where

$$A = \alpha^\star(i', j-1) \cdot \beta(i, j) + \alpha(i', j-1) \cdot \beta^\star(i, j) \quad (21)$$

$$B = 1\{i = y_j^\star\} \cdot \alpha(i', j-1) \cdot \beta(i, j) \cdot Q'_j(\mathbf{w}). \quad (22)$$

Thus, the algorithm above computes the gradient in $O(|\mathcal{Y}|^2 \cdot L)$ time and $O(|\mathcal{Y}| \cdot L)$ space. Since $\alpha^\star(i, j)$ and $\beta^\star(i, j)$ must be computed for both $\mathbf{y}^* = \mathbf{y}^{(t)}$ and $\mathbf{y}^* = \tilde{\mathbf{y}}^{(t)}$, the resulting total gradient computation takes approximately three times as long and uses twice the memory of the analogous computation for the log likelihood gradient.[5]

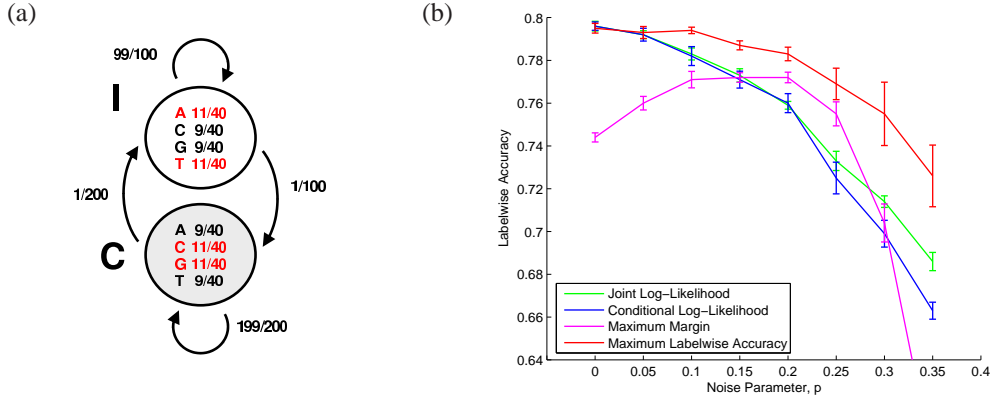

Figure 1: Panel (a) shows the state diagram for the hidden Markov model used for the simulation experiments. The HMM consists of two states ('C' and 'I') with transition probabilities labeled on the arrows, and emission probabilities specified (over the alphabet $\{A, C, G, T\}$) written inside each state. Panel (b) shows the proportion of state labels correctly predicted by the learned models at varying levels of label noise. The error bars show 95% confidence intervals on the mean generalization performance.

## 4 Results

### 4.1 Simulation experiments

To test the performance of the approximate labelwise accuracy objective function, we first ran simulation experiments in order to assess the robustness of several different learning algorithms in problems with a high degree of label noise. In particular, we generated sequences of length 1,000,000 from a simple two-state hidden Markov model (see Figure 1a). Given a fixed noise parameter $p \in [0, 1]$, we generated training sequence labels by flipping each run of consecutive 'C' hidden state labels to 'I' with probability $p$. After learning parameters, we then tested each algorithm on uncorrupted testing sequence generated by the original HMM.

Figure 1b indicates the proportion of labels correctly identified by four different methods at varying noise levels: a generative model trained with joint log-likelihood, a CRF trained with conditional log-likelihood, the maximum-margin method of Taskar et al. [9] as implemented in the SVMstruct package [10][6], and a CRF trained with maximum labelwise accuracy. No method outperforms maximum labelwise accuracy at any noise level. For levels of noise above 0.05, maximum labelwise accuracy performs significantly better than the other methods.

For each method, we used the decoding algorithm (Viterbi or MEA) that led to the best performance. The maximum margin method performed best when Viterbi decoding was used, while the other three methods had better performance with MEA decoding. Interestingly, with no noise present, maximum margin training with Viterbi decoding peformed significantly better than generative training with Viterbi decoding (0.749 vs. 0.710), but this was still much worse than generative training with MEA decoding (0.796).

### 4.2 Gene prediction experiments

To test the performance of maximum labelwise accuracy training on a large-scale, real world problem, we trained a CRF to predict protein coding genes in the genome of the fruit fly *Drosophila melanogaster*. The CRF labeled each base pair of a DNA sequence according to its predicted functional category: intergenic, protein coding, or intronic. The features used in the model were of two types: transitions between labels and trimer composition.

The CRF was trained on approximately 28 million base pairs labeled according to annotations from the FlyBase database [12]. The predictions were evaluated on a separate testing set of the same size. Three separate training runs were performed, using three different objective functions: maximum

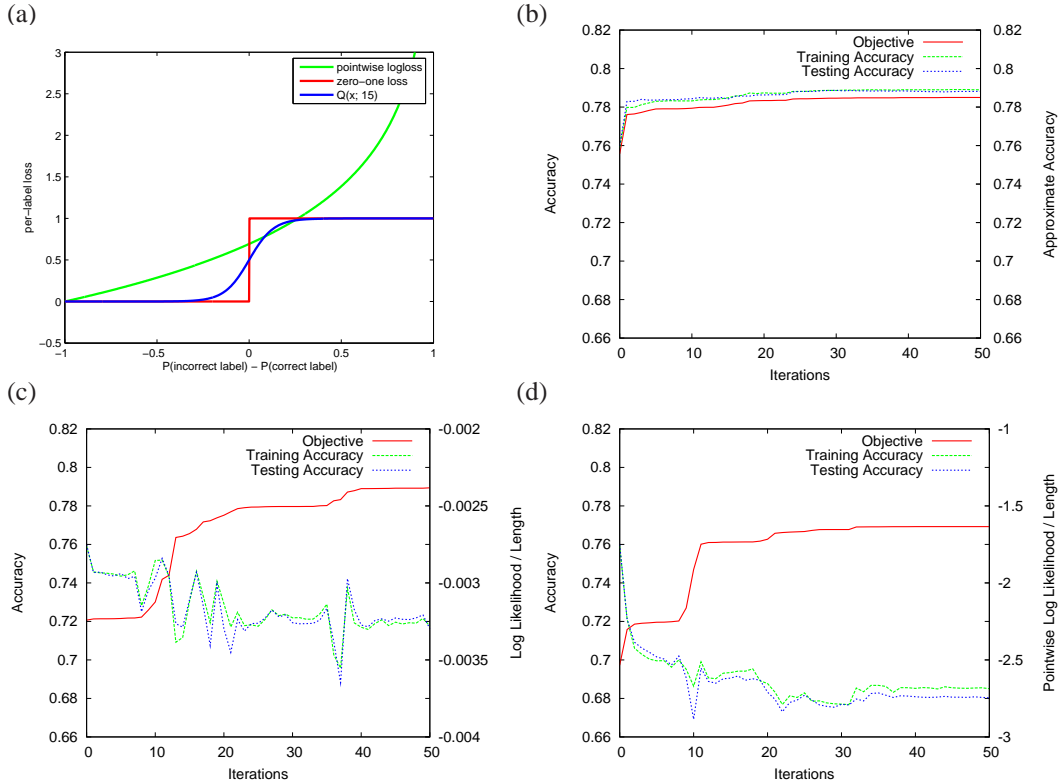

Figure 2: Panel (a) compares three pointwise loss functions in the special case where a label has two possible values. The green curve ($f(x) = -\log(\frac{1-x}{2})$) depicts pointwise logloss; the red curve represents the ideal zero-one loss; and the blue curve gives the sigmoid approximation with parameter 15. Panels (b), (c), and (d) show gene prediction learning curves using three training objective functions: (b) maximum labelwise (approximate) accuracy, (c) maximum conditional log-likelihood, and (d) maximum pointwise conditional log-likelihood, respetively. In each case, parameters were initialized to their generative model estimates.

likelihood, maximum pointwise likelihood, and maximum labelwise accuracy. Each run was started from an initial guess calculated using HMM-style generative parameter estimation.[7]

Figures 2b, 2c, and 2d show the value of the objective function and the average label accuracy at each iteration of the three training runs. Here, maximum accuracy training improves upon the accuracy of the original generative parameters and outperforms the other two training objectives. In contrast, maximum likelihood training and maximum pointwise likelihood training both give worse performance than the simple generative parameter estimates. Evidently, for this problem the likelihood-based functions are poor surrogate measures for per-label accuracy: Figures 2c and 2d show declines in training and testing set accuracy, despite increases in the objective function.

## 5    Discussion and related work

In contrast to most previous work describing alternative objective functions for CRFs, the method described in this paper optimizes a direct approximation of the Hamming loss. A few notable papers have also dealt with the problem of minimizing empirical risk directly. For binary classifiers, Jansche showed that an algorithm designed to optimize F-measure performance of a logistic regression model for information extraction outperforms maximum likelihood training [14]. For parsing tasks, Och demonstrated that a statistical machine translation system choosing between a small finite collection of candidate parses achieves better accuracy when it is trained to minimize error rate instead

of optimizing the more traditional maximum mutual information criterion [15]. Unlike Och's algorithm, our method does not require one to provide a small set of candidate parses, instead relying on efficient dynamic programming recurrences for all computations.

After this work was submitted for consideration, a Minimum Classification Error (MCE) method for training CRFs to minimize empirical risk was independently proposed by Suzuki et al. [11]. This technique minimizes the loss incurred by maximum *a posteriori*, rather than maximum expected accuracy, parsing on the training set. In practice, Viterbi parsers often achieve worse per-label accuracy than maximum expected accuracy parsers [3, 4, 5]; we are currently exploring whether a similar relationship also exists between MCE methods and our proposed training objective.

The training method described in this work is theoretically attractive, as it addresses the goal of empirical risk minimization in a very direct way. In addition to its theoretical appeal, we have shown that it performs much better than maximum likelihood and maximum pointwise likelihood training on a large scale, real world problem. Furthermore, our method is efficient, having time complexity approximately three times that of maximum likelihood likelihood training, and easily parallelizable, as each training example can be considered independently when evaluating the objective function or its gradient. The chief disadvantage of our formulation is its nonconvexity. In practice, this can be combatted by initializing the optimization with a parameter vector obtained by a convex training method. At present, the extent of the effectiveness of our method and the characteristics of problems for which it performs well are not clear. Further work applying our method to a variety of sequence labeling tasks is needed to investigate these questions.

## 6 Acknowledgments

SSG and CBD were supported by NDSEG fellowships. We thank Andrew Ng for useful discussions.

## Footnotes

[1] The gradient of the optimization objective is everywhere zero (except at points where the objective is discontinuous), because a sufficiently small change in parameters will not change the predicted labeling.

[2]We assume for simplicity the existence of a special initial label $y_0$.

[3]In particular, note that that the method of using $Q(x; \lambda)$ to approximate the step function is analogous to the log-barrier method used in convex optimization for approximating inequality constraints using a smooth function as a surrogate for the infinite height barrier. As with log-barrier optimization, performing the maximization of $\mathcal{R}_{\text{labelwise}}(\mathbf{w} : \mathcal{D})$ using a small value of $\lambda$, and gradually increasing $\lambda$ while using the previous solution as a starting point for the new optimization, provides a viable technique for maximizing the labelwise accuracy objective.

[4]Technically, the max function is not differentiable. One could replace the max with a softmax function, and assuming unique probabilities for each candidate label, the gradient approaches (11) as the softmax function approaches the max. As noted in [11], this approximation used here does not cause problems in practice.

[5]We note that the "trick" used in the formulation of approximate accuracy is applicable to a variety of other forms and arguments for $Q(\cdot)$. In particular, if we change its argument to $P_{\mathbf{w}}(y_j^{(t)} \mid \mathbf{x}^{(t)})$, letting $Q(x) = \log(x)$ gives the pointwise logloss formulation of Kakade et al. (see section 3.1.2), while letting $Q(x) = x$ gives an objective function equal to expected accuracy. Computing the gradient for these objectives involves straightforward modifications of the recurrences presented here.

[6]We were unable to get SVMstruct to converge on our test problem when using the Tsochantaridis et al. maximum margin formulation.

[7]We did not include maximum margin methods in this comparison; existing software packages for maximum margin training, based on the cutting plane algorithm [10] or decomposition techniques such as SMO [9, 13], are not easily parallelizable and scale poorly for large datasets, such as those encountered in gene prediction.

## References

[1] J. Lafferty, A. McCallum, and F. Pereira. Conditional random fields: probabilistic models for segmenting and labeling sequence data. In *ICML*, 2001.

[2] V. Vapnik. *Statistical Learning Theory*. Wiley, 1998.

[3] C. B. Do, M. S. P. Mahabhashyam, M. Brudno, and S. Batzoglou. ProbCons: probabilistic consistency-based multiple sequence alignment. *Genome Research*, 15(2):330–340, 2005.

[4] C. B. Do, D. A. Woods, and S. Batzoglou. CONTRAfold: RNA secondary structure prediction without physics-based models. *Bioinformatics*, 22(14):e90–e98, 2006.

[5] P. Liang, B. Taskar, and D. Klein. Alignment by agreement. In *HLT-NAACL*, 2006.

[6] J. Nocedal and S. J. Wright. *Numerical Optimization*. Springer, 1999.

[7] S. Kakade, Y. W. Teh, and S. Roweis. An alternate objective function for Markovian fields. In *ICML*, 2002.

[8] Y. Altun, M. Johnson, and T. Hofmann. Investigating loss functions and optimization methods for discriminative learning of label sequences. In *EMNLP*, 2003.

[9] B. Taskar, C. Guestrin, and D. Koller. Max margin markov networks. In *NIPS*, 2003.

[10] I. Tsochantaridis, T. Hofmann, T. Joachims, and Y. Altun. Support vector machine learning for interdependent and structured output spaces. In *ICML*, 2004.

[11] J. Suzuki, E. McDermott, and H. Isozaki. Training conditional random fields with multivariate evaluation measures. In *ACL*, 2006.

[12] G. Grumbling, V. Strelets, and The Flybase Consortium. FlyBase: anatomical data, images and queries. *Nucleic Acids Research*, 34:D484–D488, 2006.

[13] J. Platt. Using sparseness and analytic QP to speed training of support vector machines. In *NIPS*, 1999.

[14] M. Jansche. Maximum expected F-measure training of logistic regression models. In *EMNLP*, 2005.

[15] F. J. Och. Minimum error rate training in statistical machine translation. In *ACL*, 2003.
